# VLSI Implementation of TInMANN

**Matt Melton    Tan Phan    Doug Reeves    Dave Van den Bout**
Electrical and Computer Engineering Dept.
North Carolina State University
Raleigh, NC 27695-7911

## Abstract

A massively parallel, all-digital, stochastic architecture — TInMANN — is described which performs competitive and Kohonen types of learning. A VLSI design is shown for a TInMANN neuron which fits within a small, inexpensive MOSIS TinyChip frame, yet which can be used to build larger networks of several hundred neurons. The neuron operates at a speed of 15 MHz which allows the network to process 290,000 training examples per second. Use of level sensitive scan logic provides the chip with 100% fault coverage, permitting very reliable neural systems to be built.

## 1   INTRODUCTION

Uniprocessor simulation of neural networks has been the norm, but benefiting from the parallelism in neural networks is impossible without specialized hardware. Most hardware-based neural network simulators use a single high-speed ALU or multiple DSP chips connected through communication buses. The first approach does not allow exploration of the effects of parallelism, while the complex processors used in the second approach hinder investigations into the minimal hardware needs of an implementation. Such knowledge can be gained only if an implementation possess the same characteristics as a neural network — i.e. that it be built from many simple, cooperating processing elements. However, constructing and connecting large numbers of processing elements (or *neurons*) is difficult. Highly-connected, densely-packed analog neurons can be practically realized on a single VLSI chip, but interconnecting several such chips into a larger system would require many I/O pins. In addition, external parasitic capacitances and noise can affect the reliable transfer of data between the chips. These problems are avoided in neural systems

based on noise-resistant digital signals that can be multiplexed over a small number of wires.

The next section of this paper describes the basic theory, algorithm, and architecture of the TInMANN digital neural network. The third section illustrates the VLSI design of a TInMANN neuron that operates at 15 MHz, is completely testable, and can be cascaded to form large Kohonen or competitive networks.

## 2  TInMANN ALGORITHM AND ARCHITECTURE

In the competitive learning algorithm (Rumelhart, 1986), training vectors of length $W$, $\mathbf{v} = (v_1, v_2, \ldots, v_W)$, are presented to a winner-take-all network of $N$ neurons. Each neuron $i$ possesses a weight vector of length $W$, $\mathbf{w}_i = (w_{i1}, w_{i2}, \ldots, w_{iW})$, and a winning neuron $k$ is selected as the one whose weight vector is closest to the current training vector. Neuron $k$ is then moved closer to the training vector by modifying its weights as follows

$$w_{kj} \Leftarrow w_{kj} + \epsilon \cdot (v_j - w_{kj})  \quad 0 < \epsilon < 1, \ 1 \leq j \leq W .$$

If the network is trained with a set of vectors that are naturally clustered into $N$ groups, then each neural weight vector will eventually reside in the center of a different group. Thereafter, an input vector applied to the network is encoded by the neuron that has been sensitized to the cluster containing the input.

Kohonen's self-organizing feature maps (Kohonen, 1982) are trained using a generalization of competitive learning where each neuron $i$ is provided with an additional $X$-element vector, $\mathbf{x}_i = (x_{i1}, x_{i2}, \ldots, x_{iX})$, that defines its topological position with relation to the other neurons in the network. As before, neuron $k$ of the $N$ neurons wins if it is the closest to the current training vector, but the weight adjustment now affects *all* neurons as determined by a decreasing function $f$ of their topological distance from neuron $k$ and a threshold distance $d_T$:

$$w_{ij} \Leftarrow w_{ij} + \epsilon \cdot f(\|\mathbf{x}_k - \mathbf{x}_i\|, d_T) \cdot (v_j - w_{ij})  \quad 0 < \epsilon < 1, \ 1 \leq j \leq W, \ 1 \leq i \leq N .$$

This function allows the winning neuron to *drag* its neighbors toward a given section of the input space so that topologically close neurons will eventually react similarly to closely spaced input vectors.

The integer Markovian learning algorithm of Figure 1 simplifies the Kohonen learning procedure by noting that the neuron weights slowly *integrate* the effects of stimuli. This integration can be done by stochastically updating the weights with a probability proportional to the neural input. The stochastic update of the neural weights is done by generating two uncorrelated random numbers, $R_1$ and $R_2$, on the interval $[0, d_T]$ that each neuron compares to its distance from the current training vector and its topological distance from the winning neuron, respectively. A neuron will try to increment or decrement the elements of its weight vector closer to the training vector if the absolute value of the intervening distance is greater than $R_1$, thus creating a total movement proportional to the distance when averaged over many cycles. This movement is inversely modulated by the topological distance to the winning neuron $k$ via a comparison with $R_2$. The total effect produced by these two stochastic processes is equivalent to that produced in Kohonen's original algorithm, but only simple additive operations are now needed. Figure 2 shows

```
for( i ⇐ 1; i ≤ N; i ⇐ i+1 )
        for( j ⇐ 1; j ≤ W; j ⇐ j+1 )
                w_ij ⇐ random()
for( v∈ {training set} )
        parallelfor( all neurons i )
                d_i ⇐ c_i
                for( j ⇐ 1; j ≤ W; j ⇐ j+1 )
                        d_i ⇐ d_i + |v_j − w_ij|
        k ⇐ 1
        for( i ⇐ 1; i ≤ N; i ⇐ i+1 )
                if( d_i < d_k )
                        k ⇐ i
        parallelfor( all neurons i )
                d_i ⇐ 0
                for( j ⇐ 1; j ≤ X; j ⇐ j+1 )
                        d_i ⇐ d_i + |x_ij − x_kj|
        for( j ⇐ 1; j ≤ W; j ⇐ j+1 )
                R_1 ⇐ random(d_T)
                R_2 ⇐ random(d_T)
                parallelfor( all neurons i )
                        /* stochastic weight update */
                        if( |v_j − w_ij| > R_1 and d_i ≤ R_2 )
                                w_ij ⇐ w_ij+ sign(v_j − w_ij)
        d_T ⇐ α_d · d_T
```

**Figure 1:** The integer Markovian learning algorithm.

our simplified algorithm operates correctly on a problem that has often been solved using Kohonen networks.

The integer Markovian learning algorithm is practical to implement since only simple neurons are needed to do the additive operations and a single global bus can handle all the broadcast transmissions. The high-level architecture for such an implementation is shown in Figure 3. TInMANN consists of a global controller that coordinates the actions of a linear array of neurons. The neurons contain circuitry for comparing and updating their weights, and for enabling and disabling themselves during the conditional portions of the algorithm. The network topology is configured by arranging the neurons in an $X$-dimensional space rather than by storing a graph structure in the hardware. This allows the calculation of the topological distance between neurons using the same circuitry as is used in the weight calculations. TInMANN performs the following operations for each training vector:

1. The global controller broadcasts the $W$ elements of **v** while each neuron accumulates in $A$ the absolute value of the difference between the elements of its weight vector (stored in the small, local RAM) and those of the training vector.

2. The global controller does a binary search for the neuron closest to the training

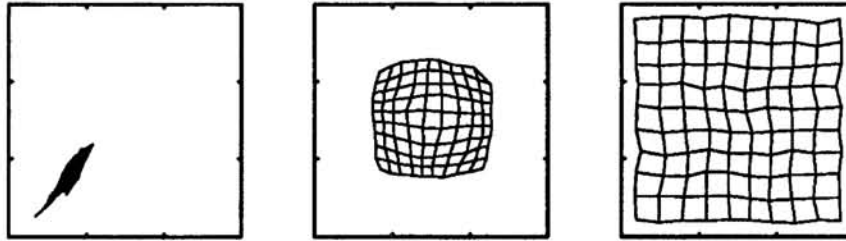

**Figure 2:** The evolution of 100 TInMANN neurons when learning a two-dimensional vector quantization.

vector by broadcasting distance values bisecting the range containing the winning neuron. The neurons do a comparison and signal on the wired-OR status line if their distance is less than the broadcast value (i.e. the carry bit $c$ is set). Neurons with distances greater than the broadcast value are disabled by resetting their $e$ flags. However, if no neuron is left enabled, the controller restores the enable bits and adjusts its search region (this action is needed on $\approx M/2$ of the search steps, where $M$ is the machine word length used by TInMANN). The last neuron left enabled is the winner of the competition (ties are resolved by the conditional logic in each neuron).

3. The topological vector of the winning neuron is broadcast to the other neurons through gate $G$. The other neurons accumulate into $A$ and store into $T_1$ the absolute value of the difference between their topological vectors and that of the winning neuron.

4. Random number $R_2$ is broadcast by the global controller and those neurons having topological distances in $T_1$ greater than $R_2$ are disabled. The remaining neurons each compute the distance between a component of their weight vector and that of the training vector broadcast by the global controller. All neurons whose calculated distances are greater than random number $R_1$ broadcast by the controller will increment or decrement their weight elements depending on the carry bits left in the $c$ flags during the distance calculations. Then all neurons are re-enabled and this step is repeated for the remaining $W - 1$ elements of the training vector.

A single training vector can be processed in $11W + X + 2.5M + 15$ clock cycles (Van den Bout, 1989). A word-width of 10 bits and a clock cycle of 15 MHz would allow TInMANN to learn at a rate of 200,000 three-dimensional vectors per second or 290,000 one-dimensional vectors per second.

## 3    THE VLSI IMPLEMENTATION OF TInMANN

Figure 4 is a block diagram for the VLSI TInMANN neuron built from the components listed in Table 1. The design was driven by the following requirements:

**Size:** The TInMANN neuron had to fit within a MOSIS TinyChip frame, so we used small, dense, ripple-carry adders. A 10-bit word size was selected as a

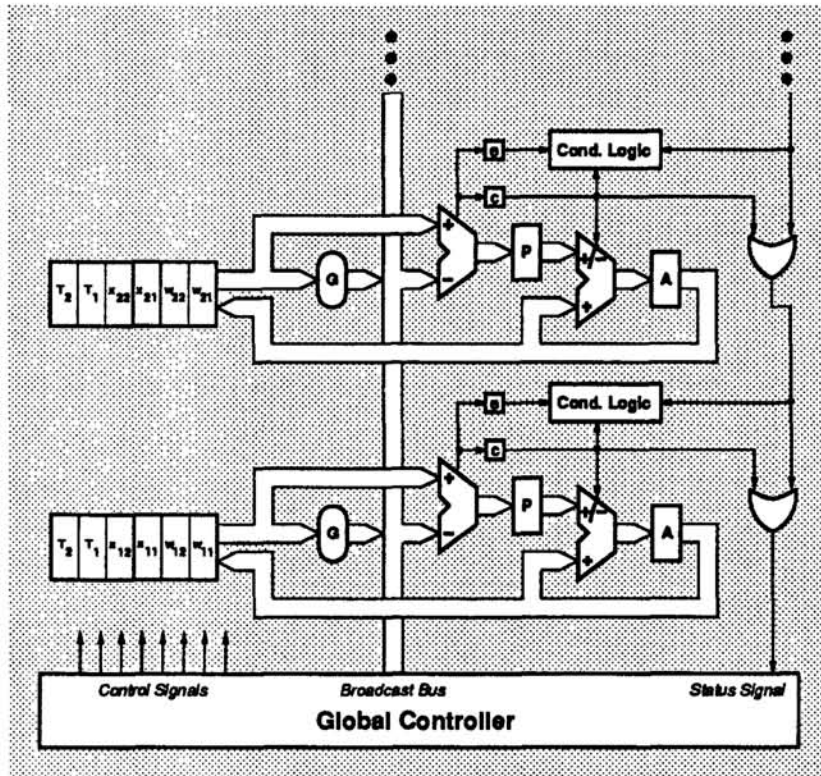

**Figure 3:** The TInMANN architecture.

**Table 1:** Components of the VLSI TInMANN neuron.

| Component | Function |
|---|---|
| ABDiff | 10-bit, two's-complement, ripple-borrow subtractor that calculates differences between data in the neuron and data broadcast on the global bus (B_Bus). |
| P | 10-bit pipeline register that temporarily stores the difference output by ABDiff. |
| CFLAG | Records the sign bit of the difference stored in P. |
| PASum | 10-bit, two's-complement, ripple-carry adder/subtractor that adds or subtracts P from the accumulator depending on the sign bit in CFLAG. This implements the absolute value function. |
| A | Accumulates the absolute values from PASum to form the Manhattan distance between a neuron and a training vector. |
| 8-word memory | Stores the weight and topology vectors, the *conscience* register (De-Sieno, 1988), and one working register. |
| MUX | Steers the output of A or the memory to the input of ABDiff. |
| EFLAG | Stores the enable bit used for conditionally controlling the neuron function during the binary search and weight update phases. |

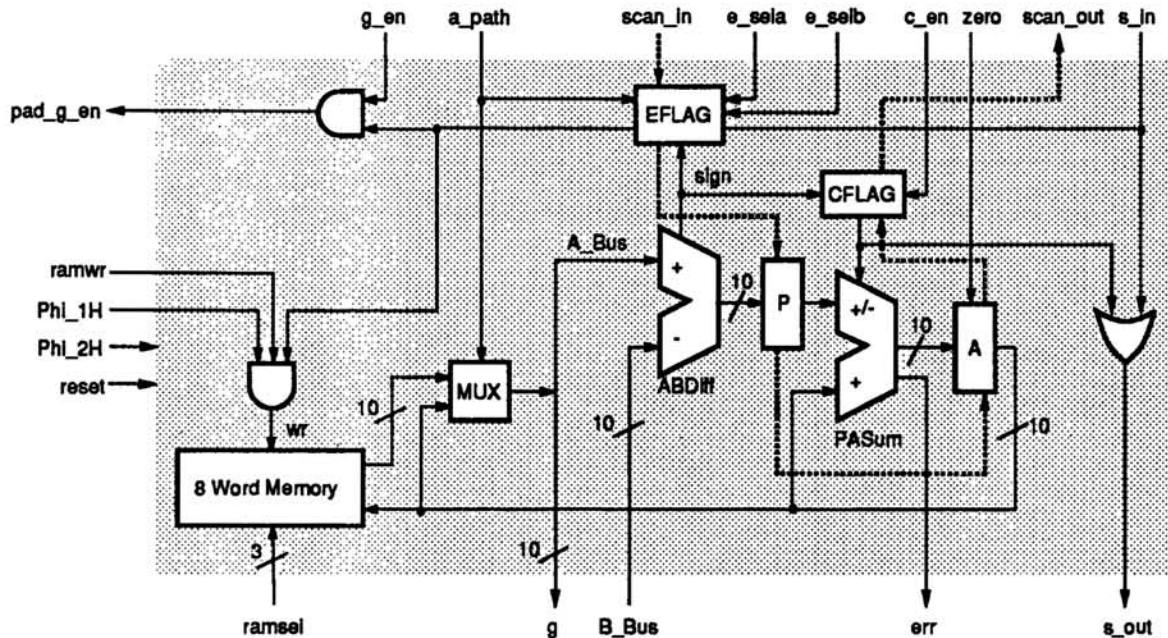

**Figure 4:** Block Diagram of the VLSI TInMANN neuron.

compromise between saving area and retaining numeric precision. The multiplexer was added so that A could be used as another temporary register. The neuron logic was built with the OASIS silicon compiler (Kedem, 1990), but the memory was hand-crafted to reduce its area. In the final TInMANN neuron, 4000 transistors are divided between the arithmetic logic ($770\mu \times 1300\mu$) and the memory ($710\mu \times 1160\mu$).

**Expandability:** The use of broadcast communications reduces the total TInMANN chip I/O to only 35 pins. This low connectivity makes it practical to build large Kohonen networks. At the chip level, the use of a silicon compiler lets us expand the design if more silicon area becomes available. For example, the word-size could be readily expanded and the layout automatically regenerated by changing a single-statement in the hardware description. Also, higher-dimensional vector spaces could be supported by adding more memory.

**Speed:** In the worst case, the memory access time is 12 ns, each adder delay is 45 ns, and the write time for A is 10 ns. This would have limited TInMANN to a top speed of 9 MHz. P was added to break the critical path through the adders and bring the clock frequency to 15 MHz. At the board level, the ripple of status information through the OR gates is sped up by connecting the status lines through an OR-tree.

**Testability:** To speed the diagnosis of system failures caused by defective chips, the TInMANN neuron was made 100% testable by building EFLAG, CFLAG, P, and A from level-sensitive scannable latches. Test patterns are shifted into the chip through the scan_in pin and the results are shifted out through scan_out. All faults are covered by only 27 test patterns. A 100% testable neural system is built by concatenating the scan_in and scan_out pins of all the chips.

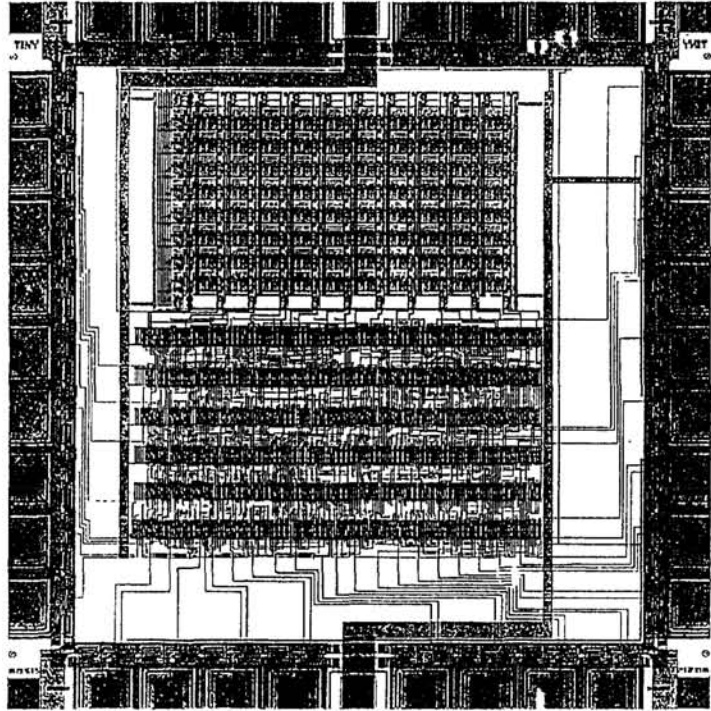

**Figure 5:**  Layout of the TInMANN neuron.

Each component of the TInMANN neuron was extensively simulated to check for correct operation. To test the chip I/O, we performed a detailed circuit simulation of two TInMANN neurons organized as a competitive network. The simulation demonstrated the movement of the two neurons towards the centroids of two data clusters used to provide training vectors.

Four of the TInMANN neurons in Figure 5 were fabricated by MOSIS. Using the built-in scan path, each was found to function at 20 MHz (the maximum speed of our tester). These chips are now being connected into a linear neural array and attached to a global controller.

## References

**D. E. Van den Bout** and **T. K. Miller III**. "TInMANN: The Integer Markovian Artificial Neural Network". In *IJCNN*, pages II:205–II:211, 1989.

**D. DeSieno.** "Adding a Conscience to Competitive Learning". In *IEEE International Conference on Neural Networks*, pages I:117–I:124, 1988.

**G. Kedem, F. Brglez,** and **K. Kozminski.** "OASIS: A Silicon Compiler for Rapid Implementation of Semi-custom Designs". In *International Workshop on Rapid Systems Prototyping*, June 1990.

**T. Kohonen.** "Self-Organized Formation of Topologically Correct Feature Maps". *Biological Cybernetics*, 43:56–69, 1982.

**D. Rumelhart** and **J. McClelland.** *Parallel Distributed Processing: Explorations in the Microstructure of Cognition*, chapter 5. MIT Press, 1986.
